# A Convergent Form of Approximate Policy Iteration

**Theodore J. Perkins**
Department of Computer Science
University of Massachusetts Amherst
Amherst, MA 01003
perkins@cs.umass.edu

**Doina Precup**
School of Computer Science
McGill University
Montreal, Quebec, Canada H3A 2A7
dprecup@cs.mcgill.ca

## Abstract

We study a new, model-free form of approximate policy iteration which uses Sarsa updates with linear state-action value function approximation for policy evaluation, and a "policy improvement operator" to generate a new policy based on the learned state-action values. We prove that if the policy improvement operator produces $\epsilon$-soft policies and is Lipschitz continuous in the action values, with a constant that is not too large, then the approximate policy iteration algorithm converges to a unique solution from any initial policy. To our knowledge, this is the first convergence result for any form of approximate policy iteration under similar computational-resource assumptions.

## 1   Introduction

In recent years, methods for reinforcement learning control based on approximating value functions have come under fire for their poor, or poorly-understood, convergence properties. With tabular storage of state or state-action values, algorithms such as Real-Time Dynamic Programming, Q-Learning, and Sarsa [2, 13] are known to converge to optimal values. Far fewer results exist for the case in which value functions are approximated using generalizing function approximators, such as state-aggregators, linear approximators, or neural networks. Arguably, the best successes of the field were generated in this way (e.g., [15]), and there are a few positive convergence results, particularly for the case of linear approximators [16, 7, 8]. However, simple examples demonstrate that many standard reinforcement learning algorithms, such as Q-Learning, Sarsa, and approximate policy iteration, can diverge or cycle without converging when combined with generalizing function approximators (e.g., [1, 6, 4]).

One classical explanation for this lack of convergence is that, even if one assumes that the agent's environment is Markovian, the problem is non-Markovian from the agent's point of view—the state features and/or the agent's approximator architecture may conspire to make some environment states indistinguishable. We focus on a more recent observation, which faults the discontinuity of the action selection strategies usually employed by reinforcement learning agents [5, 10]. If an agent uses almost any kind of generalizing function approximator to estimate state-values or state-action values, the values that are learned depend on the visitation frequencies of different states or state-action pairs. If the agent's behavior is discontinuous in its value estimates, as is the case with greedy and $\epsilon$-greedy behavior [14], then slight changes in value estimates may result in radical changes in the

agent's behavior. This can dramatically change the relative frequencies of different states or state-action pairs, causing entirely different value estimates to be learned.

One way to avoid this problem is to ensure that small changes in action values result in small changes in the agent's behavior—that is, to make the agent's policy a continuous function of its values. De Farias and Van Roy [5] showed that a form of approximate value iteration which relies on linear value function approximations and *softmax* policy improvement is guaranteed to possess fixed points. For partially-observable Markov decision processes, Perkins and Pendrith [10] showed that observation-action values that are fixed points under Q-Learning or Sarsa update rules are guaranteed to exist if the agent uses any continuous action selection strategy. Both of these papers demonstrate that continuity of the agent's action selection strategy leads to the existence of fixed points to which the algorithms can converge. In neither case, however, was convergence established.

We take this line of reasoning on step further. We study a form of approximate policy iteration in which, at each iteration: (1) Sarsa updating is used to learn weights for a linear approximation to the action value function of the current policy (policy evaluation), and then (2) a "policy improvement operator" determines a new policy based on the learned action values (policy improvement).[1] We show that if the policy improvement operator, analogous to the action selection strategy of an on-line agent, is *$\epsilon$-soft* and *Lipschitz continuous* in the action values, with a constant that is not too large, then the sequence of policies generated is guaranteed to converge. This technical requirement formalizes the intuition that the agent's behavior should not change too dramatically when value estimates change.

## 2 Markov Decision Processes and Value Functions

We consider infinite-horizon discounted Markov decision problems [3]. We assume that the Markov decision process has a finite state set, $S$, and a finite action set, $A$, with sizes $m = |S|$ and $n = |A|$. When the process is in state $s$ and the agent chooses action $a$, the agent receives an immediate reward with expectation $r_s^a$, and the process transitions to next state $s'$ with probability $p_{s,s'}^a$. Let $\mathbf{r}$ be the length $mn$ vector of expected immediate rewards following each state-action pair ($\mathbf{r}(s,a) = r_s^a$).

A stochastic policy, $\pi$, assigns a probability distribution over $A$ to each $s \in S$. The probability that the agent chooses action $a$ when the process is in state $s$ is denoted $\pi(s,a)$. If $\pi$ is deterministic in state $s$, i.e., if $\pi(s,a) = 1$ for some $a$ and $\pi(s,a') = 0$ for all $a' \neq a$, then we write $\pi(s) = a$. For $t = 0, 1, 2, \ldots$ let $s_t$, $a_t$, and $r_t$ denote, respectively, the state of the process at time $t$, the action chosen by the agent at time $t$, and the reward received by the agent at time $t$. For policy $\pi$, the state-value function, $V^\pi$, and state-action value function (or just action-value function), $Q^\pi$, are defined as:

$$V^\pi(s) = E^\pi \left\{ \sum_{t=0}^\infty \gamma^t r_t | s_0 = s \right\}, \quad Q^\pi(s,a) = E^\pi \left\{ \sum_{t=0}^\infty \gamma^t r_t | s_0 = s, a_0 = a \right\},$$

where the expectation is with respect to the stochasticity of the process and the fact that the agent chooses actions according to $\pi$, and $\gamma \in [0,1)$ is a discount factor. It is well-known [11] that there exists at least one deterministic, optimal policy $\pi^*$ for which $Q^{\pi^*}(s,a) \geq Q^\pi(s,a)$ for all $s$, $a$, and $\pi$.

Policy $\pi$ is called $\epsilon$-soft if $\pi(s,a) \geq \epsilon$ for all $s$ and $a$. For any $\epsilon > 0$, let $\Pi_\epsilon$ denote the set of $\epsilon$-soft policies. Note that a policy, $\pi$, can be viewed as an element of $\mathbb{R}^{mn}$, and $\Pi_\epsilon$ can be viewed as a compact subset of $\mathbb{R}^{mn}$. We make the following assumption:

**Assumption 1** *Under any policy $\pi$, the Markov decision process behaves as an irreducible, aperiodic Markov chain over the state set $S$.*

**Inputs:** initial policy $\pi_0$, and policy improvement operator $\Gamma$.

---

  **for** i=0,1,2,... **do**

    **Policy evaluation:** *Sarsa updates under policy $\pi_i$, with linear function approximation.*
    Initialize $\mathbf{w}_i \in \mathbb{R}^k$ arbitrarily.
    With environment in state $s_0$:
        Choose $a_0$ according to $\pi_i(s_0, \cdot)$.
        Observe $r_0, s_1$.
    Repeat for $t = 1, 2, 3, \ldots$ until $\mathbf{w}_i$ converges:
        Choose $a_t$ according to $\pi_i(s_t, \cdot)$.
        $\mathbf{w}_i \leftarrow \mathbf{w}_i + \alpha_t \Phi(s_{t-1}, a_{t-1})(r_{t-1} + \gamma \Phi'(s_t, a_t)\mathbf{w}_i - \Phi'(s_{t-1}, a_{t-1})\mathbf{w}_i)$
        Observe $r_t, s_{t+1}$.

    **Policy improvement:**
    $\pi_{i+1} \leftarrow \Gamma(\Phi\mathbf{w}_i)$.
  **end for**

---

Figure 1: The version of approximate policy iteration that we study.

The approximate policy iteration algorithm we propose learns linear approximations to the action value functions of policies. For this purpose, we assume that each state-action pair $(s, a)$ is represented by a length $k$ feature vector $\Phi(s, a)$. (In this paper, all vectors are columns unless transposed.) For weights $\mathbf{w} \in \mathbb{R}^k$, the approximate action-value for $(s, a)$ is $\hat{Q}(s, a) = \Phi'(s, a)\mathbf{w}$, where $\Phi'(s, a)$ denotes the transpose of $\Phi(s, a)$. Letting $\Phi$ be the $mn$-by-$k$ matrix whose rows correspond to the feature vectors of the state-action pairs, the entire approximate action-value function given by weights $\mathbf{w}$ is represented by the vector $\hat{Q} = \Phi\mathbf{w}$. We make the following assumption:

**Assumption 2** *The columns of $\Phi$ are linearly independent.*

## 3 Approximate Policy Iteration

The standard, exact policy iteration algorithm [3] starts with an arbitrary policy $\pi_0$ and alternates between two steps: policy evaluation, in which $V^{\pi_i}$ is computed, and policy improvement, in which a new policy, $\pi_{i+1}$, is computed. $V^{\pi_i}$ can be computed in various ways, including dynamic programming or solving a system of linear equations. $\pi_{i+1}$ is taken to be a greedy, deterministic policy with respect to $Q^{\pi_i}$. That is, $\pi_{i+1}(s) \in \arg\max_a Q^{\pi_i}(s, a) = \arg\max_a r_s^a + \sum_{s'} p_{s,s'}^a V^\pi(s')$ for all $s$. Policy iteration terminates when $V^{\pi_{i+1}} = V^{\pi_i}$. It is well-known that the sequence of policies generated is monotonically improving in the sense that $V^{\pi_{i+1}}(s) \geq V^{\pi_i}(s)$ for all $s$, and that the algorithm terminates after a finite number of iterations [3].

Bertsekas and Tsitsiklis [4] describe several versions of approximate policy iteration in which the policy evaluation step is not exact. Instead, $V^{\pi_i}$ is approximated by a weighted linear combination of state features, with weights determined by Monte Carlo or TD($\lambda$) learning rules. However, they assume that the policy improvement step is the same as in the standard policy iteration algorithm—the next policy is greedy with respect to the (approximate) action values of the previous policy. Bertsekas and Tsitsiklis show that if the approximation error in the evaluation step is low, then such algorithms generate solutions that are near optimal [4]. However, they also demonstrate by example that the sequence of policies generated does not converge for some problems, and that poor performance can result when the approximation error is high.

We study the version of approximate policy iteration shown in Figure 1. Like the versions studied by Bertsekas and Tsitsiklis, we assume that policy evaluation is not performed exactly. In particular, we assume that Sarsa updating is used to learn the weights of a linear approximation to the action-value function. We use action-value functions instead of state-value functions so that the algorithm can be performed based on interactive experience with the environment, without knowledge of the state transition probabilities. The weights learned in the policy evaluation step converge under conditions specified by Tsitsiklis and Van Roy [17], one of which is Assumption 2.

The key difference from previous work is that we assume a generic policy improvement operator, $\Gamma$, which maps every $Q \in \mathbb{R}^{mn}$ to a stochastic policy. This operator may produce, for example, greedy policies, $\epsilon$-greedy policies, or policies with action selection probabilities based on the softmax function [14]. $\Gamma$ is *Lipschitz continuous with constant c* if, for all $Q_1, Q_2 \in \mathbb{R}^{mn}$, $\|\Gamma(Q_1) - \Gamma(Q_2)\| \leq c\|Q_1 - Q_2\|$, where $\|\cdot\|$ denotes the Euclidean norm. $\Gamma$ is *$\epsilon$-soft* if, for all $Q \in \mathbb{R}^{mn}$, $\Gamma(Q)$ is $\epsilon$-soft. The fact that we allow for a policy improvement step that is not strictly greedy enables us to establish the following theorem.

**Theorem 1** *For any infinite-horizon Markov decision process satisfying Assumption 1, and for any $\epsilon > 0$, there exists $c > 0$ such that if $\Gamma$ is $\epsilon$-soft and Lipschitz continuous with constant c, then the sequence of policies generated by the approximate policy iteration algorithm in Figure 1 converges to a unique limiting policy $\pi \in \Pi_\epsilon$, regardless of the choice of $\pi_0$.*

In other words, if the behavior of the agent does not change too greatly in response to changes in its action value estimates, then convergence is guaranteed. The remainder of the paper is dedicated to proving this theorem. First, however, we briefly consider what the theorem means and what some of its limitations are. The strength of the theorem is that it states a simple condition under which a form of model-free reinforcement learning control based on approximating value functions converges for a general class of problems. The theorem does not specify a particular constant, $c$, which ensures convergence; it merely states that such a constant exists. The values of $c$ (and hence, range of policy improvement operators) which ensure convergence depend on properties of the decision process, such as its transition probabilities and rewards, which we assume to be unknown. The theorem also offers no guarantee on the quality of the policy to which the algorithm converges. Intuitively, if the policy improvement operator is Lipschitz continuous with a small constant $c$, then the agent is limited in the extent to which it can optimize its behavior. For example, even if an agent correctly learns that the value of action $a$ is much higher than the value of action $a'$, $c$ limits the frequency with which the agent can choose $a$ in favor of $a'$, and this may limit performance. The practical importance of these considerations remains to be seen, and is discussed further in the conclusions section.

## 4 Proof of Theorem 1

### 4.1 Probabilities Related to State-Action Pairs

Because the approximate policy iteration algorithm in Figure 1 approximates action-values, our analysis relies extensively on certain probabilities that are associated with state-action pairs. First, we define $P^\pi$ to be the $mn$-by-$mn$ matrix whose entries correspond to the probabilities that one state-action pair follows another when the agent behaves according to $\pi$. That is, the element on the $(s, a)^{th}$ row and $(s', a')^{th}$ column of $P^\pi$ is $p^a_{s,s'}\pi(s', a')$. $P^\pi$ can be viewed as the stochastic transition matrix of a Markov chain over state-action pairs.

**Lemma 1** *There exists $c_P$ such that for all $\pi_1, \pi_2$, $\|P^{\pi_1} - P^{\pi_2}\| \leq c_P\|\pi_1 - \pi_2\|$.*

**Proof:** Let $\pi_1$ and $\pi_2$ be fixed, and let $i = (s, a)$ and $j = (s', a')$. Then $|P^{\pi_1}_{i,j} - P^{\pi_2}_{i,j}| = |p^a_{s,s'}(\pi_1(s', a') - \pi_2(s', a'))| \leq |\pi_1(s', a') - \pi_2(s', a')| \leq \max_{s',a'} |\pi_1(s', a') - \pi_2(s', a')| = \|\pi_1 - \pi_2\|_\infty \leq \|\pi_1 - \pi_2\|$. It is readily shown that for any two $l$-by-$l$ matrices

$A$ and $B$ whose elements different in absolute value by at most $\epsilon$, $\|A - B\| \leq \sqrt{l}\epsilon$. Hence, $\|P^{\pi_1} - P^{\pi_2}\| \leq \sqrt{mn}\|\pi_1 - \pi_2\|$. $\square$

Under Assumption 1, fixing a policy, $\pi$, induces an irreducible, aperiodic Markov chain over $S$. Let $p^\pi(s) > 0$ denote the stationary probability of state $s$. We define $\mu^\pi$ to be the length $mn$ vector whose $(s, a)^{th}$ element is $p^\pi(s)\pi(s, a)$. Note that the elements of $\mu^\pi$ sum to one. If $\pi(s, a) > 0$ for all $s$ and $a$, then all elements of $\mu^\pi$ are positive and it is easily verified that $\mu^\pi$ is the unique stationary distribution of the irreducible, aperiodic Markov chain over state-action pairs with transition matrix $P^\pi$.

**Lemma 2** *For any $\epsilon > 0$, there exists $c_\mu$ such that for all $\pi_1, \pi_2 \in \Pi_\epsilon$, $\|\mu^{\pi_1} - \mu^{\pi_2}\| \leq c_\mu\|\pi_1 - \pi_2\|$.*

**Proof:** For any $\pi \in \Pi_\epsilon$, let $\lambda^\pi$ be the largest eigenvalue of $P^\pi$ with modulus strictly less than 1. $\lambda^\pi$ is well-defined since the transition matrix of any irreducible, aperiodic Markov chain has precisely one eigenvalue equal to one [11]. Since the eigenvalues of a matrix are continuous in the elements of the matrix [9], and since $\Pi_\epsilon$ is compact, there exists $\lambda^{\max} = \max_{\pi \in \Pi_\epsilon} \lambda^\pi = \lambda^{\pi_{\max}} < 1$ for some $\pi_{\max} \in \Pi_\epsilon$. Seneta [12], showed that for any two irreducible aperiodic Markov chains with transition matrices $P^1$ and $P^2$ and stationary distributions $\mu^1$ and $\mu^2$, on a state set with $l$ elements, $\|\mu^1 - \mu^2\|_1 \leq \frac{l}{|1-\lambda^1|}\|P^1 - P^2\|_\infty$, where $\lambda^1$ is the largest eigenvalue of $P^1$ with modulus strictly less than one. Let $\pi_1, \pi_2 \in \Pi_\epsilon$. $\|\mu^{\pi_1} - \mu^{\pi_2}\| \leq \|\mu^{\pi_1} - \mu^{\pi_2}\|_1 \leq \frac{mn}{|1-\lambda^{\pi_1}|}\|P^{\pi_1} - P^{\pi_2}\|_\infty \leq \frac{mn}{|1-\lambda^{\max}|}\|P^{\pi_1} - P^{\pi_2}\| \leq \frac{mn}{|1-\lambda^{\max}|}c_P\|\pi_1 - \pi_2\|$. $\square$

Lastly, we define $D^\pi$ to be the matrix whose diagonal is $\mu^\pi$. It is easy to show that for any $\pi_1, \pi_2$, $\|D^{\pi_1} - D^{\pi_2}\| \leq \|\mu^{\pi_1} - \mu^{\pi_2}\| \leq c_\mu\|\pi_1 - \pi_2\|$.

## 4.2 The Weights Learned in the Policy Evaluation Step

Consider the approximate policy evaluation step of the algorithm in Figure 1. Suppose that the agent follows policy $\pi$ and uses Sarsa updates to learn weights $\mathbf{w}$, and suppose that $\pi(s, a) > 0$ for all $s$ and $a$. Then $P^\pi$ is the stochastic transition matrix of an irreducible, aperiodic Markov chain over state-action pairs, and $D^\pi$ has the unique stationary distribution of that chain on its diagonal. Under standard conditions on the learning rate parameters for the updates, $\alpha_t$, Tsitsiklis and Van Roy [17] show that the weights converge to the unique solution to the equation:

$$\Phi'D^\pi(I - \gamma P^\pi)\Phi\mathbf{w} = \Phi'D^\pi\mathbf{r} \tag{1}$$

(Note that we have translated their result for TD($\lambda$) updating of approximate state-values to Sarsa, or TD(0), updating of approximate state-action values.) In essence, this equation says that the "expected update" to the weights under the stationary distribution, $\mu^\pi$, is zero. Let $A^\pi = \Phi'D^\pi(I - \gamma P^\pi)\Phi$ and $b^\pi = \Phi'D^\pi\mathbf{r}$. Tsitsiklis and Van Roy [17] show that $A^\pi$ is invertible, hence we can write $\mathbf{w}^\pi = (A^\pi)^{-1}b^\pi$ for the unique weights which satisfy Equation 1.

**Lemma 3** *There exist $c_b$ and $c_A$ such that for all $\pi_1, \pi_2$, $\|b^{\pi_1} - b^{\pi_2}\| \leq c_b\|\pi_1 - \pi_2\|$ and $\|A^{\pi_1} - A^{\pi_2}\| \leq c_A\|\pi_1 - \pi_2\|$.*

**Proof:** For the first claim, $\|b^{\pi_1} - b^{\pi_2}\| = \|\Phi'(D^{\pi_1} - D^{\pi_2})\mathbf{r}\| \leq \|\Phi'\|\|D^{\pi_1} - D^{\pi_2}\|\|\mathbf{r}\| \leq c_\mu\|\Phi'\|\|\mathbf{r}\|\|\pi_1 - \pi_2\|$. For the second claim,

$$
\begin{aligned}
\|A^{\pi_1} - A^{\pi_2}\| &= \|\Phi'[D^{\pi_1}(I - \gamma P^{\pi_1}) - D^{\pi_2}(I - \gamma P^{\pi_2})]\Phi\| \\
&\leq \|\Phi'\|\|D^{\pi_1}(I - \gamma P^{\pi_1}) - D^{\pi_2}(I - \gamma P^{\pi_2})\|\|\Phi\| \\
&= \|\Phi'\|\|D^{\pi_1} - D^{\pi_2} - \gamma D^{\pi_1}P^{\pi_1} + \gamma D^{\pi_2}P^{\pi_2}\|\|\Phi\| \\
&= \|\Phi'\|\|D^{\pi_1} - D^{\pi_2} - \gamma D^{\pi_1}(P^{\pi_1} - P^{\pi_2} + P^{\pi_2}) + \gamma D^{\pi_2}P^{\pi_2}\|\|\Phi\|
\end{aligned}
$$

$$\begin{aligned}
&= &&\|\Phi'\|\|D^{\pi_1} - D^{\pi_2} - \gamma D^{\pi_1}(P^{\pi_1} - P^{\pi_2}) - \gamma(D^{\pi_1} - D^{\pi_2})P^{\pi_2})\|\|\Phi\| \\
&\leq &&\|\Phi'\|(\|D^{\pi_1} - D^{\pi_2}\| + \gamma\|D^{\pi_1}\|\|P^{\pi_1} - P^{\pi_2}\| + \gamma\|D^{\pi_1} - D^{\pi_2}\|\|P^{\pi_2}\|)\|\Phi\| \\
&\leq &&((1+\gamma)c_\mu + \gamma c_P)\|\Phi'\|\|\Phi\|\|\pi_1 - \pi_2\| \,,
\end{aligned}$$

where the last line follows from Lemmas 1 and 2 and the facts $\|D^\pi\| \leq 1$ and $\|P^\pi\| = 1$ for any $\pi \in \Pi_\epsilon$. $\square$

**Lemma 4** *For any $\epsilon > 0$, there exists $c_w$ such that $\|\mathbf{w}^\pi\| \leq c_w$ for all $\pi \in \Pi_\epsilon$.*

**Proof:** By Lemmas 1 and 2, and by the continuity of matrix inverses [11], $\mathbf{w}^\pi$ is a continuous function of $\pi$. Thus, $\|\mathbf{w}^\pi\|$ is a continuous function of $\pi$. Because $\Pi_\epsilon$ is a compact subset of $\mathbb{R}^{mn}$, and because continuous functions map compact sets to compact sets, the existence of the bound, $c_w$, follows. $\square$

For any $mn$-by-$mn$ matrix $M$, let $g(M) = \min_{\|x\|=1}\|Mx\|$. That is, $g(M)$ measures how small a vector of length one can become under left-multiplication by matrix $M$.

**Lemma 5** *For any $\epsilon > 0$, there exists $c_g > 0$ such that for all $\pi \in \Pi_\epsilon$, $g(A^\pi) \geq c_g$.*

**Proof:** Lemma 7, in the Appendix, shows that $g$ is a continuous mapping and that $g$ is positive for any non-singular matrix. For any $\pi \in \Pi_\epsilon$, $g(A^\pi) \geq \inf_{\pi_1 \in \Pi_\epsilon} g(A^{\pi_1})$. Since $g$ is continuous, and $\Pi_\epsilon$ compact, the infimum is attained by some $\pi_{\inf} \in \Pi_\epsilon$. Thus $g(A^\pi) \geq g(A^{\pi_{\inf}}) > 0$, where the last inequality follows because $A^{\pi_{\inf}}$ is non-singular. $\square$

**Lemma 6** *For any $\epsilon > 0$, there exists $c_{w2}$ such that for all $\pi_1, \pi_2 \in \Pi_\epsilon$, $\|\mathbf{w}^{\pi_1} - \mathbf{w}^{\pi_2}\| \leq c_{w2}\|\pi_1 - \pi_2\|$.*

**Proof:** Let $\pi_1, \pi_2 \in \Pi_\epsilon$ be arbitrary. From Equation 1, $A^{\pi_1}\mathbf{w}^{\pi_1} = b^{\pi_1}$ and $A^{\pi_2}\mathbf{w}^{\pi_2} = b^{\pi_2}$. Thus:

$$\begin{aligned}
& A^{\pi_1}\mathbf{w}^{\pi_1} - A^{\pi_2}\mathbf{w}^{\pi_2} &&= && b^{\pi_1} - b^{\pi_2} \\
\Rightarrow\ & A^{\pi_1}(\mathbf{w}^{\pi_1} - \mathbf{w}^{\pi_2} + \mathbf{w}^{\pi_2}) - A^{\pi_2}\mathbf{w}^{\pi_2} &&= && b^{\pi_1} - b^{\pi_2} \\
\Rightarrow\ & A^{\pi_1}(\mathbf{w}^{\pi_1} - \mathbf{w}^{\pi_2}) + (A^{\pi_1} - A^{\pi_2})\mathbf{w}^{\pi_2} &&= && b^{\pi_1} - b^{\pi_2} \\
\Rightarrow\ & A^{\pi_1}(\mathbf{w}^{\pi_1} - \mathbf{w}^{\pi_2}) &&= && (b^{\pi_1} - b^{\pi_2}) - (A^{\pi_1} - A^{\pi_2})\mathbf{w}^{\pi_2} \\
\Rightarrow\ & \|A^{\pi_1}(\mathbf{w}^{\pi_1} - \mathbf{w}^{\pi_2})\| &&\leq && \|b^{\pi_1} - b^{\pi_2}\| + \|A^{\pi_1} - A^{\pi_2}\|\|\mathbf{w}^{\pi_2}\| \\
\Rightarrow\ & c_g\|\mathbf{w}^{\pi_1} - \mathbf{w}^{\pi_2}\| &&\leq && c_b\|\pi_1 - \pi_2\| + c_w c_A\|\pi_1 - \pi_2\| \quad (2) \\
\Rightarrow\ & \|\mathbf{w}^{\pi_1} - \mathbf{w}^{\pi_2}\| &&\leq && c_g^{-1}(c_b + c_w c_A)\|\pi_1 - \pi_2\|
\end{aligned}$$

The left hand side of Equation 2 follows from Lemmas 5 and 7; the right hand side follows from Lemmas 3 and 4. $\square$

## 4.3 Contraction Argument

**Proof of Theorem 1:** For a given infinite-horizon discounted Markov decision problem, let $\epsilon > 0$ and $\Gamma$ be fixed. Suppose that $\Gamma$ is Lipschitz continuous with constant $c$, where $c$ is yet to be determined. Let $\pi_1, \pi_2 \in \Pi_\epsilon$ be arbitrary. The policies that result from $\pi_1$ and $\pi_2$ after one iteration of the approximate policy iteration algorithm of Figure 1 are $\Gamma(\hat{Q}^{\pi_1})$ and $\Gamma(\hat{Q}^{\pi_2})$ respectively. Observe that: $\|\Gamma(\hat{Q}^{\pi_1}) - \Gamma(\hat{Q}^{\pi_2})\| \leq c\|\hat{Q}^{\pi_1} - \hat{Q}^{\pi_2}\| = c\|\Phi(\mathbf{w}^{\pi_1} - \mathbf{w}^{\pi_2})\| \leq c\|\Phi\|c_{w2}\|\pi_1 - \pi_2\|$, where the last step follows from Lemma 6. If $c < \|\Phi\|^{-1}c_{w2}^{-1}$, then for some $\beta \in [0, 1)$ we have $\|\Gamma(\hat{Q}^{\pi_1}) - \Gamma(\hat{Q}^{\pi_2})\| \leq \beta\|\pi_1 - \pi_2\|$. Each iteration of the approximate policy iteration is a contraction. By the Contraction Mapping Theorem [3], there is a unique fixed point of the mapping $\pi \to \Gamma(\hat{Q}^\pi)$, and the sequence of policies generated according to that mapping from any initial policy converges to the fixed point. $\square$

Note that since the sequence of policies, $\pi_i$, converges, and since $\hat{Q}^\pi$ is a continuous function of $\pi$, the sequence of approximate action-value functions computed by the algorithm, $\hat{Q}^{\pi_i}$, also converges.

# 5 Conclusions and Future Work

We described a model-free, approximate version of policy iteration for infinite-horizon discounted Markov decision problems. In this algorithm, the policy evaluation step of classical policy iteration is replaced by learning a linear approximation to the action-value function using on-line Sarsa updating. The policy improvement step is given by an arbitrary policy improvement operator, which maps any possible action-value function to a new policy. The main contribution of the paper is to show that if the policy improvement operator is $\epsilon$-soft and Lipschitz continuous in the action-values, with a constant $c$ that is not too large, then the approximate policy iteration algorithm is guaranteed to converge to a unique, limiting policy from any initial policy. We are hopeful that similar ideas can be used to establish the convergence of other reinforcement learning algorithms, such as on-line Sarsa or Sarsa($\lambda$) control with linear function approximation.

The magnitude of the constant $c$ that ensures convergence depends on the model of the environment and on properties of the feature representation. If the model is not known, then choosing a policy improvement operator that guarantees convergence is not immediate. To be safe, an operator for which $c$ is small should be chosen. However, one generally prefers $c$ to be large, so that the agent can exploit its knowledge by choosing actions with higher estimated action-values as frequently as possible. One approach to determining a proper value of $c$ would be to make an initial guess and begin the approximate policy iteration procedure. If the contraction property fails on any iteration, one should choose a new policy improvement operator that is Lipschitz continuous with a smaller constant. A potential advantage of this approach is that one can begin with a high choice of $c$, which allows exploitation of action value differences, and switch to lower values of $c$ only as necessary. It is possible that convergence could be obtained with much higher values of $c$ than are suggested by the bound in the proof of Theorem 1.

Discontinuous improvement operators/action selection strategies can lead to non-convergent behavior for many reinforcement learning algorithms, including Q-Learning, Sarsa, and forms of approximate policy iteration and approximate value iteration. For some of these algorithms, (non-unique) fixed points have been shown to exist when the action selection strategy/improvement operator is continuous [5, 10]. Whether or not convergence also follows remains to be seen. For the algorithm studied in this paper, we have constructed an example demonstrating non-convergence with improvement operators that are Lipschitz continuous but with too large of a constant. In this case, it appears that the Lipschitz continuity assumption we use cannot be weakened. One direction for future work is determining minimal restrictions on action selection (if any) that ensure the convergence of other reinforcement learning algorithms.

Ensuring convergence answers one standing objection to reinforcement learning control methods based on approximating value functions. However, an important open issue for our approach, and for other approaches advocating continuous action selection [5, 10], is to characterize the solutions that they produce. We know of no theoretical guarantees on the quality of solutions found, and there is little experimental work comparing algorithms that use continuous action selection with those that do not.

**Acknowledgments**
Theodore Perkins was supported in part by National Science Foundation grants ECS-0070102 and ECS-9980062. Doina Precup was supported in part by grants from NSERC and FQNRT.

## Footnotes

[1]The algorithm can also be viewed as batch-mode Sarsa with linear action-value function approximation.

# References

[1] L. C. Baird. Residual algorithms: Reinforcement learning with function approximation. In *Proceedings of the Twelfth International Conference on Machine Learning*, pages 30–37. Morgan Kaufmann, 1995.

[2] A. G. Barto, S. J. Bradtke, and S. P. Singh. Learning to act using real-time dynamic programming. *Artificial Intelligence*, 72(1):81–138, 1995.

[3] D. P. Bertsekas. *Dynamic Programming and Optimal Control, Volumes 1 and 2*. Athena Scientific, 2001.

[4] D. P. Bertsekas and J. N. Tsitsiklis. *Neuro-Dynamic Programming*. Athena Scientific, 1996.

[5] D. P. De Farias and B. Van Roy. On the existence of fixed points for approximate value iteration and temporal-difference learning. *Journal of Opt. Theory and Applications*, 105(3), 2000.

[6] G. Gordon. Chattering in Sarsa($\lambda$). CMU Learning Lab Internal Report. Available at www.cs.cmu.edu/~ggordon, 1996.

[7] G. Gordon. *Approximate Solutions to Markov Decision Processes*. PhD thesis, Carnegie Mellon University, 1999.

[8] G. J. Gordon. Reinforcement learning with function approximation converges to a region. *Advances in Neural Information Processing Systems 13*, pages 1040–1046. MIT Press, 2001.

[9] C. D. Meyer. *Matrix Analysis and Applied Linear Algebra*. SIAM, 2000.

[10] T. J. Perkins and M. D. Pendrith. On the existence of fixed points for Q-learning and Sarsa in partially observable domains. In *Proceedings of the Nineteenth International Conference on Machine Learning*, 2002.

[11] M. L. Puterman. *Markov Decision Processes: Disrete Stochastic Dynamic Programming*. John Wiley & Sons, Inc, New York, 1994.

[12] E. Seneta. Sensitivity analysis, ergodicity coefficients, and rank-one updates for finite markov chains. In W. J. Stewart, editor, *Numerical Solutions of Markov Chains*. Dekker, NY, 1991.

[13] S. Singh, T. Jaakkola, M. L. Littman, and C. Szepesvari. Convergence results for single-step on-policy reinforcement-learning algorithms. *Machine Learning*, 38(3):287–308, 2000.

[14] R. S. Sutton and A. G. Barto. *Reinforcement Learning: An Introduction*. MIT Press/Bradford Books, Cambridge, Massachusetts, 1998.

[15] G. J. Tesauro. TD-Gammon, a self-teaching backgammon program, achieves master-level play. *Neural Computation*, 6(2):215–219, 1994.

[16] J. N. Tsitsiklis and B. Van Roy. Optimal stopping of markov processes: Hilbert space theory, approximation algorithms, and an application to pricing high-dimensional financial derivatives. *IEEE Transactions on Automatic Control*, 44(10):1840–1851, 1999.

[17] J. N. Tsitsiklis and B. Van Roy. An analysis of temporal-difference learning with function approximation. *IEEE Transactions on Automatic Control*, 42(5):674–690, 1997.

## Appendix

**Lemma 7** *For l-by-l matrix $M$, let $g(M) = \min_{\|x\|=1} \|Mx\|$. Then:*

1. *$g(M) \geq 0$ for all $M$,*
2. *$g(M) > 0$ iff $M$ is non-singular,*
3. *for any $x \in \mathbb{R}^l$, $\|Mx\| \geq g(M)\|x\|$,*
4. *$g$ is continuous.*

**Proof:** The first three points readily follow from elementary arguments. We focus on the last point. We want to show that given a sequence of matrices $M^i$, $i \in \{1, 2, 3, \ldots\}$ that converge to some $M^*$, then $g(M^i) \to g(M^*)$. Note that $M^i \to M^*$ means that $\lim_{i \to \infty} \|M^i - M^*\| = 0$. Let $x^* \in \arg\min_{\|x\|=1} \|M^*x\|$. Then $\limsup_{i \to \infty} g(M^i) = \limsup_{i \to \infty} \min_{\|x\|=1} \|M^i x\| \leq \limsup_{i \to \infty} \|M^i x^*\| = \limsup_{i \to \infty} \|(M^i - M^* + M^*)x^*\| \leq \limsup_{i \to \infty} \|(M^i - M^*)\|\|x^*\| + \|M^*x^*\| = g(M^*)$. Now, for all $i$ let $x^i \in \arg\min_{\|x\|=1} \|M^i x\|$. Then $\liminf_{i \to \infty} g(M^i) = \liminf_{i \to \infty} \|M^i x^i\| = \liminf_{i \to \infty} \|(M^i - M^* + M^*)x^i\| \geq \liminf_{i \to \infty} \|M^*x^i\| - \|(M^i - M^*)x^i\| \geq \liminf_{i \to \infty} \|M^*x^*\| - \|M^i - M^*\|\|x^i\| = g(M^*)$. Thus, $\lim_{i \to \infty} g(M^i) = g(M^*)$. $\square$